# Combining Features for BCI

**Guido Dornhege**[1]*, **Benjamin Blankertz**[1], **Gabriel Curio**[2], **Klaus-Robert Müller**[1,3]

[1]Fraunhofer FIRST.IDA, Kekuléstr. 7, 12489 Berlin, Germany

[2]Neurophysics Group, Dept. of Neurology, Klinikum Benjamin Franklin,
Freie Universität Berlin, Hindenburgdamm 30, 12203 Berlin, Germany

[3]University of Potsdam, August-Bebel-Str. 89, 14482 Potsdam, Germany

`{dornhege,blanker,klaus}@first.fraunhofer.de, curio@zedat.fu-berlin.de`

## Abstract

Recently, interest is growing to develop an effective communication interface connecting the human brain to a computer, the 'Brain-Computer Interface' (BCI). One motivation of BCI research is to provide a new communication channel substituting normal motor output in patients with severe neuromuscular disabilities. In the last decade, various neurophysiological cortical processes, such as slow potential shifts, movement related potentials (MRPs) or event-related desynchronization (ERD) of spontaneous EEG rhythms, were shown to be suitable for BCI, and, consequently, different independent approaches of extracting BCI-relevant EEG-features for single-trial analysis are under investigation. Here, we present and systematically compare several concepts for combining such EEG-features to improve the single-trial classification. Feature combinations are evaluated on movement imagination experiments with 3 subjects where EEG-features are based on either MRPs or ERD, or both. Those combination methods that incorporate the assumption that the single EEG-features are physiologically mutually independent outperform the plain method of 'adding' evidence where the single-feature vectors are simply concatenated. These results strengthen the hypothesis that MRP and ERD reflect at least partially independent aspects of cortical processes and open a new perspective to boost BCI effectiveness.

## 1 Introduction

A brain-computer interface (BCI) is a system which translates a subject's intentions into a control signal for a device, e.g., a computer application, a wheelchair or a neuroprosthesis, cf. [1]. When measuring non-invasively, brain activity is acquired by scalp-recorded electroencephalogram (EEG) from a subject that tries to convey its intentions by behaving according to well-defined paradigms, e.g., motor imagery, specific mental tasks, or feedback control. 'Features' (or feature vectors) are extracted from the digitized EEG-signals by signal processing methods. These features are translated into a control signal, either (1) by simple equations or threshold criteria (with only a few free parameters that are estimated on training data), or (2) by machine learning algorithms that learn a more complex

decision function on the training data, e.g., linear discriminant analysis (LDA), support vector machines (SVMs), or artificial neural networks (ANN).

Concerning the pivotal step of feature extraction, neurophysiological a priori knowledge can aid to decide which EEG-feature is to be expected to hold the most discriminative information for the chosen paradigm. For some behavioral paradigms even several EEG-features might be usable, stimulating a discussion how to combine different features. Investigations in this direction were announced, e.g., in [2, 3] but no publications on that topic followed.

Here, we present several methods for combining features to enhance single-trial EEG classification for BCI. A special focus was placed on the question how to incorporate a priori knowledge about feature independence. Recently this approach proved to be most effective in an open internet-based classification competition: it turned out winning entry of the NIPS BCI competition 2001, dataset 2, cf. `http://newton.bme.columbia.edu/competitionresults.htm`.

**Neurophysiological background for single-feature EEG-paradigms.**

Three approaches are characteristic for the majority of single-feature BCI paradigms. (1) Based on slow cortical potentials the Tübinger Thought Translation Device (TTD) [4] translates low-pass filtered brain activity from central scalp position into a vertical cursor movement on a computer screen. This enables subjects to learn self-regulation of electro-cortical positivity or negativity. After some training, patients can generate binary decisions in a 4 seconds pace with an accuracies of up to 85 % and thereby handle a word processor or an internet browser. (2) The Albany BCI system [2] allows the user to control cursor movement by oscillatory brain activity into one of two or four possible target areas on a computer screen. In the first training sessions most subjects use some kind of motor imagery which is replaced by adapted strategies during further feedback sessions. Well-trained users achieve hit rates of over 90 % in the two-target setup. Each selection typically takes 4 to 5 seconds. And (3), the Graz BCI system [5] is based on event-related modulations of the pericentral $\mu$- and/or $\beta$-rhythms of sensorimotor cortices, with a focus on motor preparation and imagination. Feature vectors calculated from spontaneous EEG signals by adaptive auto-regressive modelling are used to train a classifier. In a ternary classification task accuracies of over 96 % were obtained in an offline study with a trial duration of 8 seconds.

**Neurophysiological background for combining single EEG-features.**

Most gain from a combination of different features is expected when the single features provide complementary information for the classification task. In the case of movement related potentials (MRPs) or event-related desynchronization (ERD) of EEG rhythms, recent evidence [6] supports the hypothesis that MRPs and ERD of the pericentral alpha rhythm reflect different aspects of sensorimotor cortical processes and could provide complementary information on brain activity accompanying finger movements, as they show different spatiotemporal activation patterns, e.g., in primary (sensori-)motor cortex (M-1), supplementary motor area (SMA) and posterior parietal cortex (PP).

This hypothesis is backed by invasive recordings [7] supporting the idea that ERD and MRPs represent different aspects of motor cortex activation with varying generation mechanisms: EEG was recorded during brisk, self-paced finger and foot movements subdurally in 3 patients and scalp-recorded in normal subjects. MRPs started over wide areas of the sensorimotor cortices (*Bereitschaftspotential*) and focalizes at the contralateral M-1 hand cortex with a steep negative slope prior to finger movement onset, reaching a negative peak approximately 100 ms after EMG onset (*motor potential*). In contrast, a bilateral M-1 ERD just prior to movement onset appeared to reflect a more widespread cortical 'alerting' function. Most importantly, the ERD response magnitude did not have a significant correlation with the amplitude of the negative MRPs slope.

Note that these studies analyze movement preparation and execution only. We presume a similar independence of MRP and ERD phenomena for imagined movements. This hy-

pothesis is confirmed by our results, see section 3.

Apart from exploiting complementary information on cortical processes, combining MRP and ERD based features might give the benefit of being more robust against artifacts from non central nervous system (CNS) activity such as eye movement (EOG) or muscular artifacts (EMG). While EOG activity mainly affects slow potentials, i.e. MRPs, EMG activity is of more concern to oscillatory features, cf. [1]. Accordingly, a classification method that is based on both features has better chance to handle trials that are contaminated by one kind of those artifacts. On the other hand, it might increase the risk of using non-CNS activity for classification which would not be conform with the BCI idea, [1]. For our setting the latter issue is investigated in section 2.3.

## 2    Data acquisition and analysis methods

**Experiments.**

In this paper we analyze EEG data from experiments with three subjects called *aa*, *af* and *ak*. The subject sat in a normal chair, with arms lying relaxed on the table. During the experiment the symbol 'L' or 'R' was shown every 4.5 ±0.25 sec for a duration of 3 s on the computer screen. The subject was instructed to imagine performing left resp. right hand finger movements as long as the symbol was visible. 200–300 trials were recorded for each class and each subject.

Brain activity was recorded with 28 (subject *aa*) resp. 52 (subjects *af* and *ak*) Ag/AgCl electrodes at 1000 Hz and downsampled to 100 Hz for the present offline study. In addition, an electromyogram (EMG) of the *musculus flexor digitorum* bilaterally and horizontal and vertical electrooculograms (EOG) were recorded to monitor non-CNS activity.

No artifact rejection or correction was employed.

**Objective of single-trial analysis.**

In these experiments the aim of classification is to discriminate 'left' from 'right' trials based on EEG-data during the whole period of imagination. Here, no effort was made to come to a decision as early as possible, which would also be a reasonable objective.

### 2.1   Feature Extraction

The present behavioural paradigms allowed to study the two prominent brain signals accompanying motor imagery: (1) the lateralized MRP showing up as a slow negative EEG-shift focussed over the corresponding motor and sensorimotor cortex contralateral to the involved hand, and (2) the ERD appearing as a lateralized attenuation of the $\mu$- and/or central $\beta$-rhythm. Fig. 1 shows these effects calculated from subject *aa*.

In the following we describe methods to derive feature vectors capturing MRP or ERD effects. Note that all filtering techniques used are causal so that all methods are applicable in online systems. Some free parameters were chosen from appropriately fixed parameter sets by cross-validation for all experiments and each classification setting separately described in section 2.2. This selection was done to obtain the most appropriate setting for each single-feature analysis. These values were used for both, classifying trials based on single-features and the combined classification.

**Movement related potential (MRP).**

To quantify the lateralized MRP we proceeded similar to our approach in [8] (Berlin Brain-Computer Interface, BBCI). Small modifications were made to take account of the different experimental setup. Signals were baseline corrected on the interval 0–300 ms and downsampled by calculating five jumping means in several consecutive intervals beginning at 300 ms and ending between 1500–3500 ms. Optional an elliptic IIR low-pass filter at 2.5 Hz

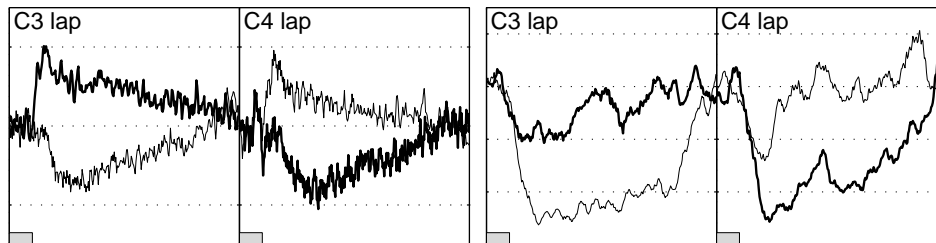

Figure 1: ERP and ERD (7–30 Hz) curves for subject *aa* in the time interval -500 ms to 3000 ms relative to stimulus. Thin and thick lines are averages over right resp. left hand trials. The contralateral negativation resp. desynchronization is clearly observable.

was applied to the signals beforehand.

To derive feature vectors for the ERD effects we use two different methods which may reflect different aspects of brain rhythm modulations. The first (AR) reflects the spectral distribution of the most prominent brain rhythms whereas the second (CSP) reflects spatial patterns of most prominent power modulation in specifying frequency bands.

**Autoregressive models (AR).**

In an autoregressive model of order $p$ each time point of a time series is represented as a fixed linear combination (AR coefficients) of the last $p$ data points. The model order $p$ was taken as free parameter to be selected between 5 and 12. The feature vector of one trial is the concatenation of the AR coefficients plus the variance of each channel. The AR coefficients reflect oscillatory properties of the EEG signal, but not the overall amplitude. Accounting for this by adding the variance to the feature vector improves classification. To prevent the AR models from being distorted by EEG-baseline drifts, the signals were high-pass filtered at 4 Hz. And to sharpen the spectral information to focal brain sources (spatial) Laplacian filters were applied. The interval for estimating the AR parameters started at 500 ms and the end points were choosen between 2000 ms and 3500 ms.

**Common spatial patterns (CSP).**

This method was suggested for binary classification of EEG trials in [9]. In features space projections on orientations with most differing power-ratios are used. These can be calculated by determining generalized eigenvalues or by simultaneous diagonalisation of the covariance matrices of both classes. Only a few orientations with the highest ratio between their eigenvalues (in both directions) are selected. The number of CSP used per class was a free parameter to be chosen between 2 and 4. Before applying CSP, the signals were filtered between 8 and 13 Hz to focus on effects in the $\alpha$-band. Using a broader band of 7–30 Hz did not give better results. The interval of interest were choosen as described above for the AR model. Feature vectors consist of the variances of the CSP projected trial, cf. [9]. Note that for cross-validation CSP must be calculated for each training set separately.

## 2.2 Classification and model selection

Our approach for classification was guided by two general ideas. First, following the concept 'simple methods first' we employed only linear classifiers. In our BCI studies linear classification methods were never found to perform worse than non-linear classifiers, cf. also [10, 11]. And second, regularization, which is a well-established principle in machine learning, is highly relevant in experimental conditions typical for a BCI scenario, i.e., a small number of training samples for 'weak features'. In weak features discriminative information is spread across many dimensions. Classifying such features based on a small training set may lead to the well-known overfitting problem. To avoid this, typically one of the following strategies is employed: (1) performing *strong preprocessing* to extract low

dimensional feature vectors which are tractable for most classifiers. Or (2) performing no or weak preprocessing and *carefully regularizing* the classifier such that high-dimensional features can be handled even with only a small training set. Solution (1) has the disadvantage that strong assumptions about the data distributions have to be made. So especially in EEG analysis where many sources of variability make strong assumptions dubious, solution (2) is to be preferred. A good introduction to regularized classification is [12] including regularized LDA which we used here.

To assess classification performance, the generalization error was estimated by $10 \times 10$-fold cross-validation. The reported standard deviation is calculated from the mean errors of the 10-fold cross-validations. The regularization coefficients were chosen by cross-validation together with the free parameters of the feature extraction methods, see section 2.1, in the following way. Strictly this cross-validation has to be performed on the training set. So in this off-line analysis where in each cross-validation procedure 100 different training sets are drawn randomly from the set of all trials one would have to do a cross-validation (for model selection, MS) within a cross-validation (for estimating the generalization error, GE). Obviously this would be very time consuming. On the other hand doing the model selection by cross-validation on all trials would could lead to overfitting and underestimating the generalization error. As an intermediate way MS-cross-validation was performed on three subsets of all trials that were randomly drawn where the size of the subsets was the same as the size of the training sets in the GE-cross-validation, i.e., here 90 % of the whole set. This procedure was tested in several settings without any significant bias on the estimation of the GE, cf. [13].

## 2.3 Analysis of single-features

The table in Fig. 2 shows the generalization error for single-features. Data of each subject can be well classified. Some differences in the quality of the features for classification are observable, but there is not one type of feature that is generally the best.

The $10 \times 10$-fold cross-validation was also used to determine how often each trial is classified correctly when belonging to the test set. Trials which were classified 9 to 10 times (i.e., 90 to 100 %) correctly are labeled 'good', while those classified 9 to 10 times wrong are labeled 'bad'. Only a small number of trials did fall in neither of those two categories ('ambivalent') as could be expected due to the small standard deviation. It is now interesting to see whether there are trials which are for one feature type in the well classified range and for the other feature in the badly classified part. Fig. 2 shows BP and CSP for subject *af* as example for each the part of the bad classified values which are good and bad classified in the other feature.

These results strengthen the hypothesis that it is promising to combine features.

We made the following check for the impact of non-CNS activity on classification results. MRP based classification was applied to the EOG signals and ERD based classification was applied to the EMG signals. All those tests resulted in accuracies at chance level ($\sim$50 %). Since the main concern in this paper is comparing classification with single vs. combined features this issue was not followed in further detail.

## 2.4 Combination methods

Feature combination or sensor fusion strategies are rather common in speech recognition (e.g. [14]) or vision (e.g. [15]) or robotics (e.g. [16]) where either signals on different time-scales or from distinct modalities need to be combined. Typical approaches suggested are a winner-takes-all strategy, which cannot increase performance above the best single feature analysis, and concatenation of the single feature vectors, discussed as CONCAT below. Furthermore combinations that use a joint probabilistic modeling [15] appear promising. We propose two further methods that incorporate independence assumptions (PROB and to

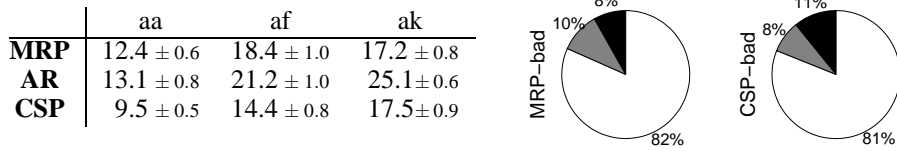

|      | aa             | af             | ak             |
|------|----------------|----------------|----------------|
| **MRP** | $12.4 \pm 0.6$ | $18.4 \pm 1.0$ | $17.2 \pm 0.8$ |
| **AR**  | $13.1 \pm 0.8$ | $21.2 \pm 1.0$ | $25.1 \pm 0.6$ |
| **CSP** | $9.5 \pm 0.5$  | $14.4 \pm 0.8$ | $17.5 \pm 0.9$ |

Figure 2: Left: Misclassification rates for single features classified with regularized LDA. Free parameters of each feature extraction method were selected by cross-validation on subsets of all trials, see section 2.2. Right: Pie charts show how 'MRP-bad' and 'CSP-bad' trials for subject *af* are classified based on the respective other feature: white is the portion of the trials which is 'good' for the other feature, black marks 'bad', and gray 'ambivalent' trials for the other feature. See text for the definition of 'good', 'bad' and 'ambivalent' in this context.

a smaller extend META) and allow individual decision boundary fitting to single features (META).

**(CONCAT)** In this simple approach of gathered evidence feature vectors are just concatenated. To account for the increased dimensionality careful regulization is necessary. Additionally, we tried classification with a linear programming machine (LPM), which is appealing for its sparse feature selection property, but it did not improve results compared to regularized LDA.

**(PROB)** It is well-known that LDA is the Bayes-optimal classifier, i.e., the one minimizing the expected risk of misclassification, for two classes of known gaussian distribution with equal covariance matrices. Here we derive the optimal classifier for combined feature vectors $X = (X_1, \ldots, X_n)$ under the additional assumption that individual features $X_1, \ldots, X_n$ are mutually independent. Denoting by $\hat{Y}(x)$ the decision function on feature space $X$

$$
\begin{aligned}
\hat{Y}(x) = \text{'R'} \quad &\Leftrightarrow \quad P(Y = \text{'R'} \mid X = x) > P(Y = \text{'L'} \mid X = x) \\
&\Leftrightarrow \quad f_{Y=\text{'R'}}(x)\, P(Y = \text{'R'}) > f_{Y=\text{'L'}}(x)\, P(Y = \text{'L'}),
\end{aligned}
$$

where $Y$ is a random variable on the labels $\{\text{'L'}, \text{'R'}\}$ and $f$ denotes densities. Using the independence assumption one can factorize the densities. Neglecting the class priors and exploiting the gaussian assumption $(X_n \mid Y = y) \sim \mathcal{N}(\mu_{n,y}, \Sigma_n)$ we get the decision function

$$
\hat{Y}(x) = \text{'R'} \quad \Leftrightarrow \quad \sum_{n=1}^{N} \left[ w_n^\top x_n - \frac{1}{2} (\mu_{n,\text{'R'}} + \mu_{n,\text{'L'}})^\top w_n \right] > 0, \ \text{with } w_n := \Sigma_n^{-1} (\mu_{n,\text{'R'}} - \mu_{n,\text{'L'}})
$$

In terms of LDA this corresponds to forcing the elements of the estimated covariance matrix that belong to different features to zero. Thereby less parameters have to be estimated and distortions by accidental correlations of independent variables are avoided. If the classes do not have equal covariance matrices a non-linear version of PROB can be formulated in analogy to quadratic discriminant analysis (QDA). To avoid overfitting we use regularisation for PROB. There are two ways possible: Regularisation of the covariance matrices with one global parameter (PROBsame) or with three separately selected parameters corresponding to the single-type features (PROBdiff).

**(META)** In this approach a meta classifier is applied to the continuous output of individual classifiers that are trained on single features beforehand. This allows a tailor-made choice of classifiers for each feature, e.g., if the decision boundary is linear for one feature and non-linear for another. Here we just use LDA for all features, but regularization coefficients are selected for each single feature individually. Since the meta classifier acts on low (2 or 3) dimensional features further regularization is not needed, so we used unregularized LDA. META extracts discriminative information from single features independently but the meta classification may exploit inter relations based on the output of the individual decision

|        | Best Single    | CONCAT         | PROBsame       | PROBdiff       | META           |
|--------|----------------|----------------|----------------|----------------|----------------|
| *aa*   | 9.5 ± 0.5      | 9.5 ± 0.4      | **6.3** ± 0.5  | 6.5 ± 0.5      | 6.7 ± 0.4      |
| *af*   | 14.4 ± 0.8     | 14.4 ± 1.2     | 7.4 ± 0.8      | **7.4** ± 0.7  | 10.2 ± 0.5     |
| *ak*   | 17.2 ± 0.8     | 14.8 ± 0.9     | 13.9 ± 1.0     | **13.2** ± 0.7 | 14.0 ± 0.8     |
| mean   | 13.7 ± 3.2     | 12.9 ± 2.4     | 9.2 ± 3.4      | 9.0 ± 3.0      | 10.3 ± 3.0     |

Table 1: Generalization errors ± s.d. of the means in 10×10-fold cross-validation for combined features compared to the most successful single-type feature. Best result for each subject is in boldface.

functions. That means independence is assumed on the low level while possible high level relations are taken into account.

## 3 Results

Table 1 shows the results for the combined classification methods and for comparison the best result on single-type features ('Best Single') from the table of Fig. 2. All three feature were combined together. Combining two of them (especially MRP with AR or CSP) leads to good values, too, which are slightly worse, however.
The CONCAT method performs only for subject *ak* better than the single feature methods. The following two problems may be responsible for that. First, there are only few training samples and a higher dimensional space than for the single features, so the curse of dimensionality stikes harder. And second, regularisation for the single features results in different regularisation parameters. In CONCAT a single regularisation parameter has to be found. In our case the regularisation parameters for subject *aa* for MRP are about 0.001 whereas for CSP about 0.8.
From the other approaches the PROB methods are most successful, but META is very good, too, and better than the single feature results. Differences between the two PROB methods were not observed.
Concerning the results it is noteworthy that all subjects were BCI-untrained. Only subject *aa* had experience as subject in EEG experiments. The result obtained with single-features is in the range of the best results for *untrained* BCI performance with imagined movement paradigm, cf. [17]. Whereas the result of less than 8 % error with our proposed combining approach for subject *aa* and *af* is better than for the 3 subjects in [17] in up to even 10 feedback sessions. Subject *ak* with an error rate of less than 14 % is in the range of good results. Additionally, it should be noted that the subject *aa* reported that he sometimes missed to react to the stimulus due to fatigue. He estimated the portion of missed stimuli to be 5 %. Hence the classification error of 6.3 % is very close to what is possible to achieve.

## 4 Concluding discussion

Combining the feature vectors corresponding to event-related desynchronization and movement-related potentials under an independence assumption derived from a priori physiological knowledge (PROB, and to a smaller extent META) leads to an improved classification accuracy when compared to single-feature classification. In contrast, the combination of features without any assumption of independence (CONCAT) did not improve accuracy in every case and always performs worse than PROB and META. These results further support the hypothesis that MRP and ERD reflect independent aspects of brain activity.
In all three experiments an improvement of about 25 % to 50 % reduction of the error rate could be achieved by combining methods. Additionally, the combined approach has the practical advantage that no prior decision has to be made about what feature to use.
Combining features of different brain processes in feedback scenarios where the subject is trying to adapt to the feedback algorithm could in principle hold the risk of making the learning task too complex for the subject. This, however, needs to be investigated in future online studies.

Finally, we would like to remark that the proposed feature combination principles can be used in other application areas where independent features can be obtained.

**Acknowledgments.**

We thank Sebastian Mika, Roman Krepki, Thorsten Zander, Gunnar Raetsch, Motoaki Kawanabe and Stefan Harmeling for helpful discussions. The studies were supported by a grant of the *Bundesministerium für Bildung und Forschung* (BMBF), FKZ 01IBB02A and FKZ 01IBB02B.

## Footnotes

*To whom correspondence should be addressed.

## References

[1] J. R. Wolpaw, N. Birbaumer, D. J. McFarland, G. Pfurtscheller, and T. M. Vaughan, "Brain-computer interfaces for communication and control", *Clin. Neurophysiol.*, 113: 767–791, 2002.

[2] J. R. Wolpaw, D. J. McFarland, and T. M. Vaughan, "Brain-Computer Interface Research at the Wadsworth Center", *IEEE Trans. Rehab. Eng.*, 8(2): 222–226, 2000.

[3] J. A. Pineda, B. Z. Allison, and A. Vankov, "The Effects of Self-Movement, Observation, and Imagination on $\mu$–Rhythms and Readiness Potential (RP's): Toward a Brain-computer Interface (BCI)", *IEEE Trans. Rehab. Eng.*, 8(2): 219–222, 2000.

[4] N. Birbaumer, N. Ghanayim, T. Hinterberger, I. Iversen, B. Kotchoubey, A. Kübler, J. Perelmouter, E. Taub, and H. Flor, "A spelling device for the paralysed", *Nature*, 398: 297–298, 1999.

[5] B. O. Peters, G. Pfurtscheller, and H. Flyvbjerg, "Automatic Differentiation of Multichannel EEG Signals", *IEEE Trans. Biomed. Eng.*, 48(1): 111–116, 2001.

[6] C. Babiloni, F. Carducci, F. Cincotti, P. M. Rossini, C. Neuper, G. Pfurtscheller, and F. Babiloni, "Human Movement-Related Potentials vs Desynchronization of EEG Alpha Rhythm: A High-Resolution EEG Study", *NeuroImage*, 10: 658–665, 1999.

[7] C. Toro, G. Deuschl, R. Thather, S. Sato, C. Kufta, and M. Hallett, "Event-related desynchronization and movement-related cortical potentials on the ECoG and EEG", *Electroencephalogr. Clin. Neurophysiol.*, 93: 380–389, 1994.

[8] B. Blankertz, G. Curio, and K.-R. Müller, "Classifying Single Trial EEG: Towards Brain Computer Interfacing", in: T. G. Diettrich, S. Becker, and Z. Ghahramani, eds., *Advances in Neural Inf. Proc. Systems (NIPS 01)*, vol. 14, 2002, to appear.

[9] H. Ramoser, J. Müller-Gerking, and G. Pfurtscheller, "Optimal spatial filtering of single trial EEG during imagined hand movement", *IEEE Trans. Rehab. Eng.*, 8(4): 441–446, 2000.

[10] L. Parra, C. Alvino, A. C. Tang, B. A. Pearlmutter, N. Yeung, A. Osman, and P. Sajda, "Linear spatial integration for single trial detection in encephalography", *NeuroImage*, 2002, to appear.

[11] K.-R. Müller, C. W. Anderson, and G. E. Birch, "Linear and Non-Linear Methods for Brain-Computer Interfaces", *IEEE Trans. Neural Sys. Rehab. Eng.*, 2003, submitted.

[12] J. H. Friedman, "Regularized Discriminant Analysis", *J. Amer. Statist. Assoc.*, 84(405): 165–175, 1989.

[13] G. Rätsch, T. Onoda, and K.-R. Müller, "Soft Margins for AdaBoost", *Machine Learning*, 42(3): 287–320, 2001.

[14] N. Morgan and H. Bourlard, "Continuous Speech Recognition: An Introduction to the Hybrid HMM/Connectionist Approach", *Signal Processing Magazine*, 25–42, 1995.

[15] M. Brand, N. Oliver, and A. Pentland, "Coupled hidden markov models for complex action recognition", 1996.

[16] S. Thrun, A. Bücken, W. Burgard, D. Fox, T. Fröhlinghaus, D. Henning, T. Hofmann, M. Krell, and T. Schmidt, "Map Learning and High-Speed Navigation in RHINO", in: D. Kortenkamp, R. Bonasso, and R. Murphy, eds., *AI-based Mobile Robots*, MIT Press, 1998.

[17] G. Pfutscheller, C. Neuper, D. Flotzinger, and M. Pregenzer, "EEG-based discrimination between imagination of right and left hand movement", *Electroencephalogr. Clin. Neurophysiol.*, 103: 642–651, 1997.
